# Learning Generative Models with the Up-Propagation Algorithm

**Jong-Hoon Oh and H. Sebastian Seung**
Bell Labs, Lucent Technologies
Murray Hill, NJ 07974
{jhoh|seung}@bell-labs.com

## Abstract

Up-propagation is an algorithm for inverting and learning neural network generative models. Sensory input is processed by inverting a model that generates patterns from hidden variables using top-down connections. The inversion process is iterative, utilizing a negative feedback loop that depends on an error signal propagated by bottom-up connections. The error signal is also used to learn the generative model from examples. The algorithm is benchmarked against principal component analysis in experiments on images of handwritten digits.

In his doctrine of unconscious inference, Helmholtz argued that perceptions are formed by the interaction of bottom-up sensory data with top-down expectations. According to one interpretation of this doctrine, perception is a procedure of sequential hypothesis testing. We propose a new algorithm, called up-propagation, that realizes this interpretation in layered neural networks. It uses top-down connections to generate hypotheses, and bottom-up connections to revise them.

It is important to understand the difference between up-propagation and its ancestor, the backpropagation algorithm[1]. Backpropagation is a learning algorithm for *recognition* models. As shown in Figure 1a, bottom-up connections recognize patterns, while top-down connections propagate an error signal that is used to learn the recognition model.

In contrast, up-propagation is an algorithm for inverting and learning *generative* models, as shown in Figure 1b. Top-down connections generate patterns from a set of hidden variables. Sensory input is processed by inverting the generative model, recovering hidden variables that could have generated the sensory data. This operation is called either pattern recognition or pattern analysis, depending on the meaning of the hidden variables. Inversion of the generative model is done iteratively, through a negative feedback loop driven by an error signal from the bottom-up connections. The error signal is also used for learning the connections

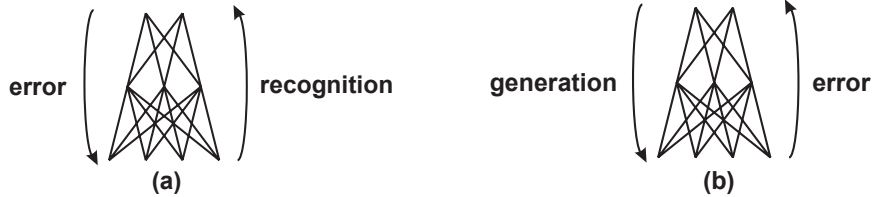

Figure 1: Bottom-up and top-down processing in neural networks. (a) Backprop network (b) Up-prop network

in the generative model.

Up-propagation can be regarded as a generalization of principal component analysis (PCA) and its variants like Conic[2] to nonlinear, multilayer generative models. Our experiments with images of handwritten digits demonstrate that up-propagation learns a global, nonlinear model of a pattern manifold. With its global parametrization, this model is distinct from locally linear models of pattern manifolds[3].

# 1  INVERTING THE GENERATIVE MODEL

The generative model is a network of $L+1$ layers of neurons, with layer 0 at the bottom and layer $L$ at the top. The vectors $x_t$, $t = 0 \ldots L$, are the activations of the layers. The pattern $x_0$ is generated from the hidden variables $x_L$ by a top-down pass through the network,

$$x_{t-1} = f(W_t x_t), \qquad t = L, \ldots, 1 .$$ (1)

The nonlinear function $f$ acts on vectors component by component. The matrix $W_t$ contains the synaptic connections from the neurons in layer $t$ to the neurons in layer $t-1$. A bias term $b_{t-1}$ can be added to the argument of $f$, but is omitted here. It is convenient to define auxiliary variables $\hat{x}_t$ by $x_t = f(\hat{x}_t)$. In terms of these auxiliary variables, the top-down pass is written as

$$\hat{x}_{t-1} = W_t f(\hat{x}_t)$$ (2)

Given a sensory input $d$, the top-down generative model can be inverted by finding hidden variables $x_L$ that generate a pattern $x_0$ matching $d$. If some of the hidden variables represent the identity of the pattern, the inversion operation is called *recognition*. Alternatively, the hidden variables may just be a more compact representation of the pattern, in which case the operation is called *analysis* or *encoding*. The inversion is done iteratively, as described below.

In the following, the operator $*$ denotes elementwise multiplication of two vectors, so that $z = x * y$ means $z_i = x_i y_i$ for all $i$. The bottom-up pass starts with the mismatch between the sensory data $d$ and the generated pattern $x_0$,

$$\delta_0 = f'(\hat{x}_0) * (d - x_0) ,$$ (3)

which is propagated upwards by

$$\delta_t = f'(\hat{x}_t) * (W_t^T \delta_{t-1}) .$$ (4)

When the error signal reaches the top of the network, it is used to update the hidden variables $x_L$,

$$\Delta x_L \propto W_L^T \delta_{L-1} .$$ (5)

This update closes the negative feedback loop. Then a new pattern $x_0$ is generated by a top-down pass (1), and the process starts over again.

This iterative inversion process performs gradient descent on the cost function $\frac{1}{2}|d - x_0|^2$, subject to the constraints (1). This can be proved using the chain rule, as in the traditional derivation of the backprop algorithm. Another method of proof is to add the equations (1) as constraints, using Lagrange multipliers,

$$\frac{1}{2}|d - f(\hat{x}_0)|^2 + \sum_{t=1}^{L} \delta_{t-1}^T [\hat{x}_{t-1} - W_t f(\hat{x}_t)] \ . \tag{6}$$

This derivation has the advantage that the bottom-up activations $\delta_t$ have an interpretation as Lagrange multipliers.

Inverting the generative model by negative feedback can be interpreted as a process of sequential hypothesis testing. The top-down connections generate a hypothesis about the sensory data. The bottom-up connections propagate an error signal that is the disagreement between the hypothesis and data. When the error signal reaches the top, it is used to generate a revised hypothesis, and the generate-test-revise cycle starts all over again. Perception is the convergence of this feedback loop to the hypothesis that is most consistent with the data.

## 2   LEARNING THE GENERATIVE MODEL

The synaptic weights $W_t$ determine the types of patterns that the network is able to generate. To learn from examples, the weights are adjusted to improve the network's generation ability. A suitable cost function for learning is the reconstruction error $\frac{1}{2}|d - x_0|^2$ averaged over an ensemble of examples. Online gradient descent with respect to the synaptic weights is performed by a learning rule of the form

$$\Delta W_t \propto \delta_{t-1} x_t^T \ . \tag{7}$$

The same error signal $\delta$ that was used to invert the generative model is also used to learn it.

The batch form of the optimization is compactly written using matrix notation. To do this, we define the matrices $D, X_0, \ldots, X_L$ whose columns are the vectors $d$, $x_0, \ldots, x_L$ corresponding to examples in the training set. Then computation and learning are the minimization of

$$\min_{X_L, W_t} \frac{1}{2}|D - X_0|^2 \ , \tag{8}$$

subject to the constraint that

$$X_{t-1} = f(W_t X_t) \ , \qquad t = 1, \ldots, L \ . \tag{9}$$

In other words, up-prop is a dual minimization with respect to hidden variables and synaptic connections. Computation minimizes with respect to the hidden variables $X_L$, and learning minimizes with respect to the synaptic weight matrices $W_t$.

From the geometric viewpoint, up-propagation is an algorithm for learning pattern manifolds. The top-down pass (1) maps an $n_L$-dimensional vector $x_L$ to an $n_0$-dimensional vector $x_0$. Thus the generative model parametrizes a continuous $n_L$-dimensional manifold embedded in $n_0$-dimensional space. Inverting the generative model is equivalent to finding the point on the manifold that is closest to the sensory data. Learning the generative model is equivalent to deforming the manifold to fit a database of examples.

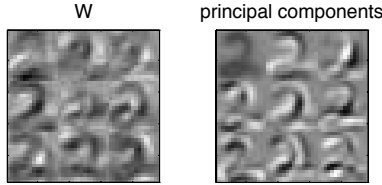

Figure 2: One-step generation of handwritten digits. Weights of the 256-9 up-prop network (left) versus the top 9 principal components (right)

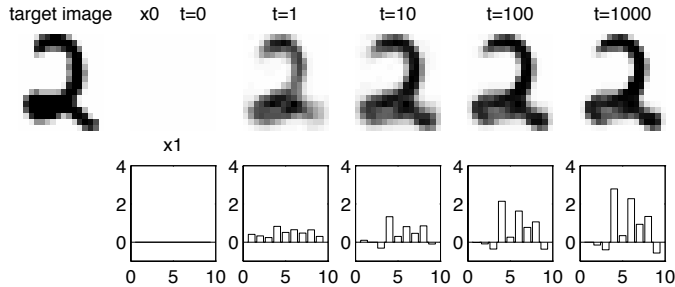

Figure 3: Iterative inversion of a generative model as sequential hypothesis testing. A fully trained 256–9 network is inverted to generate an approximation to a target image that was not previously seen during training. The stepsize of the dynamics was fixed to 0.02 to show time evolution of the system.

Pattern manifolds are relevant when patterns vary continuously. For example, the variations in the image of a three-dimensional object produced by changes of viewpoint are clearly continuous, and can be described by the action of a transformation group on a prototype pattern. Other types of variation, such as deformations in the shape of the object, are also continuous, even though they may not be readily describable in terms of transformation groups. Continuous variability is clearly not confined to visual images, but is present in many other domains. Many existing techniques for modeling pattern manifolds, such as PCA or PCA mixtures[3], depend on linear or locally linear approximations to the manifold. Up-prop constructs a globally parametrized, nonlinear manifold.

## 3   ONE-STEP GENERATION

The simplest generative model of the form (1) has just one step ($L = 1$). Up-propagation minimizes the cost function

$$\min_{X_1, W_1} \frac{1}{2} |D - f(W_1 X_1)|^2 \ . \tag{10}$$

For a linear $f$ this reduces to PCA, as the cost function is minimized when the vectors in the weight matrix $W_1$ span the same space as the top principal components of the data $D$.

Up-propagation with a one-step generative model was applied to the USPS database[4], which consists of example images of handwritten digits. Each of the 7291 training and 2007 testing images was normalized to a $16 \times 16$ grid with pixel intensities in the range $[0, 1]$. A separate model was trained for each digit class. The nonlinearity $f$ was the logistic function. Batch optimization of (10) was done by

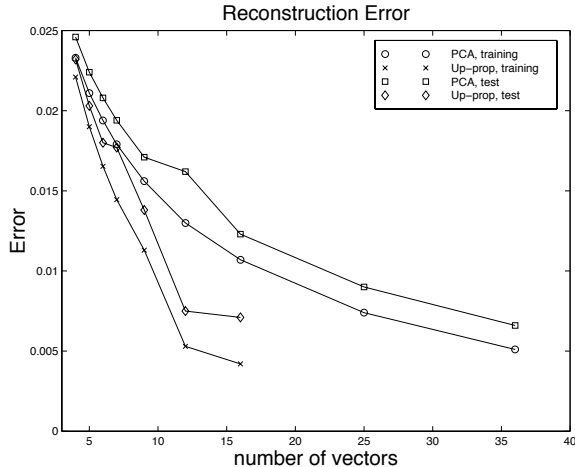

Figure 4: Reconstruction error for 256–$n$ networks as a function of $n$. The error of PCA with $n$ principal components is shown for comparison. The up-prop network performs better on both the training set and test set.

gradient descent with adaptive stepsize control by the Armijo rule[5]. In most cases, the stepsize varied between $10^{-1}$ and $10^{-3}$, and the optimization usually converged within $10^3$ epochs. Figure 2 shows the weights of a 256–9 network that was trained on 731 different images of the digit "two." Each of the 9 subimages is the weight vector of a top-level neuron. The top 9 principal components are also shown for comparison.

Figure 3 shows the time evolution of a fully trained 256–9 network during iterative inversion. The error signal from the bottom layer $x_0$ quickly activates the top layer $x_1$. At early times, all the top layer neurons have similar activation levels. However, the neurons with weight vectors more relevant to the target image become dominant soon, and the other neurons are deactivated.

The reconstruction error (10) of the up-prop network was much better than that of PCA. We trained 10 different up-prop networks, one for each digit, and these were compared with 10 corresponding PCA models. Figure 4 shows the average squared error per pixel that resulted. A 256–12 up-prop network performed as well as PCA with 36 principal components.

## 4   TWO-STEP GENERATION

Two-step generation is a richer model, and is learned using the cost function

$$\min_{X_2, W_1, W_2} \frac{1}{2} |D - f(W_1 f(W_2 X_2))|^2 \ . \tag{11}$$

Note that a nonlinear $f$ is necessary for two-step generation to have more representational power than one-step generation. When this two-step generative model was trained on the USPS database, the weight vectors in $W_1$ learned features resembling principal components. The activities of the $X_1$ neurons tended to be close to their saturated values of one or zero.

The reconstruction error of the two-step generative network was compared to that of the one-step generative network with the same number of neurons in the top layer.

Our 256–25–9 network outperformed our 256–9 network on the test set, though both networks used nine hidden variables to encode the sensory data. However, the learning time was much longer, and iterative inversion was also slow. While up-prop for one-step generation converged within several hundred epochs, up-prop for two-step generation often needed several thousand epochs or more to converge. We often found long plateaus in the learning curves, which may be due to the permutation symmetry of the network architecture[6].

## 5   DISCUSSION

To summarize the experiments discussed above, we constructed separate generative models, one for each digit class. Relative to PCA, each generative model was superior at encoding digits from its corresponding class. This enhanced generative ability was due to the use of nonlinearity.

We also tried to use these generative models for recognition. A test digit was classified by inverting all the generative models, and then choosing the one best able to generate the digit. Our tests of this recognition method were not encouraging. The nonlinearity of up-propagation tended to improve the generation ability of models corresponding to all classes, not just the model corresponding to the correct classification of the digit. Therefore the improved encoding performance did not immediately transfer to improved recognition.

We have not tried the experiment of training one generative model on all the digits, with some of the hidden variables representing the digit class. In this case, pattern recognition could be done by inverting a single generative model. It remains to be seen whether this method will work.

Iterative inversion was surprisingly fast, as shown in Figure 3, and gave solutions of surprisingly good quality in spite of potential problems with local minima, as shown in Figure 4. In spite of these virtues, iterative inversion is still a problematic method. We do not know whether it will perform well if a single generative model is trained on multiple pattern classes. Furthermore, it seems a rather indirect way of doing pattern recognition.

The up-prop generative model is deterministic, which handicaps its modeling of pattern variability. The model can be dressed up in probabilistic language by defining a prior distribution $P(x_L)$ for the hidden variables, and adding Gaussian noise to $x_0$ to generate the sensory data. However, this probabilistic appearance is only skin deep, as the sequence of transformations from $x_L$ to $x_0$ is still completely deterministic. In a truly probabilistic model, like a belief network, every layer of the generation process adds variability.

In conclusion, we briefly compare up-propagation to other algorithms and architectures.

1. In backpropagation[1], only the recognition model is explicit. Iterative gradient descent methods can be used to invert the recognition model, though this implicit generative model generally appears to be inaccurate[7, 8].

2. Up-propagation has an explicit generative model, and recognition is done by inverting the generative model. The accuracy of this implicit recognition model has not yet been tested empirically. Iterative inversion of generative models has also been proposed for linear networks[2, 9] and probabilistic belief networks[10].

3. In the autoencoder[11] and the Helmholtz machine[12], there are separate

models of recognition and generation, both explicit. Recognition uses only bottom-up connections, and generation uses only top-down connections. Neither process is iterative. Both processes can operate completely independently; they only interact during learning.

4. In attractor neural networks[13, 14] and the Boltzmann machine[15], recognition and generation are performed by the same recurrent network. Each process is iterative, and each utilizes both bottom-up and top-down connections. Computation in these networks is chiefly based on positive, rather than negative feedback.

Backprop and up-prop suffer from a lack of balance in their treatment of bottom-up and top-down processing. The autoencoder and the Helmholtz machine suffer from inability to use iterative dynamics for computation. Attractor neural networks lack these deficiencies, so there is incentive to solve the problem of learning attractors[14].

This work was supported by Bell Laboratories. JHO was partly supported by the Research Professorship of the LG-Yonam Foundation. We are grateful to Dan Lee for helpful discussions.

# References

[1] D. E. Rumelhart, G. E. Hinton, and R. J. Williams. Learning internal representations by back-propagating errors. *Nature*, 323:533–536, 1986.

[2] D. D. Lee and H. S. Seung. Unsupervised learning by convex and conic coding. *Adv. Neural Info. Proc. Syst.*, 9:515–521, 1997.

[3] G. E. Hinton, P. Dayan, and M. Revow. Modeling the manifolds of images of handwritten digits. *IEEE Trans. Neural Networks*, 8:65–74, 1997.

[4] Y. LeCun et al. Learning algorithms for classification: a comparison on handwritten digit recognition. In J.-H. Oh, C. Kwon, and S. Cho, editors, *Neural networks: the statistical mechanics perspective*, pages 261–276, Singapore, 1995. World Scientific.

[5] D. P. Bertsekas. *Nonlinear programming*. Athena Scientific, Belmont, MA, 1995.

[6] K. Kang, J.-H. Oh, C. Kwon, and Y. Park. Generalization in a two-layer neural network. *Phys. Rev.*, E48:4805–4809, 1993.

[7] J. Kindermann and A. Linden. Inversion of neural networks by gradient descent. *Parallel Computing*, 14:277–286, 1990.

[8] Y. Lee. Handwritten digit recognition using K nearest-neighbor, radial-basis function, and backpropagation neural networks. *Neural Comput.*, 3:441–449, 1991.

[9] R. P. N. Rao and D. H. Ballard. Dynamic model of visual recognition predicts neural response properties in the visual cortex. *Neural Comput.*, 9:721–63, 1997.

[10] L. K. Saul, T. Jaakkola, and M. I. Jordan. Mean field theory for sigmoid belief networks. *J. Artif. Intell. Res.*, 4:61–76, 1996.

[11] G. W. Cottrell, P. Munro, and D. Zipser. Image compression by back propagation: an example of extensional programming. In N. E. Sharkey, editor, *Models of cognition: a review of cognitive science*. Ablex, Norwood, NJ, 1989.

[12] G. E. Hinton, P. Dayan, B. J. Frey, and R. M. Neal. The "wake-sleep" algorithm for unsupervised neural networks. *Science*, 268:1158–1161, 1995.

[13] H. S. Seung. Pattern analysis and synthesis in attractor neural networks. In K.-Y. M. Wong, I. King, and D.-Y. Yeung, editors, *Theoretical Aspects of Neural Computation: A Multidisciplinary Perspective*, Singapore, 1997. Springer-Verlag.

[14] H. S. Seung. Learning continuous attractors in recurrent networks. *Adv. Neural Info. Proc. Syst.*, 11, 1998.

[15] D. H. Ackley, G. E. Hinton, and T. J. Sejnowski. A learning algorithm for Boltzmann machines. *Cognitive Science*, 9:147–169, 1985.
